# Generating velocity tuning by asymmetric recurrent connections

**Xiaohui Xie**[1] **and Martin A. Giese**[1,2]
[1]Dept. of Brain and Cognitive Sciences and CBCL
Massachusetts Institute of Technology
Cambridge, MA 02139
[2]Dept. for Cognitive Neurology,
University Clinic Tübingen
Max-Planck-Institute for Biological Cybernetics
72076 Tübingen, Germany
E-mail: {xhxie|giese}@mit.edu

## Abstract

Asymmetric lateral connections are one possible mechanism that can account for the direction selectivity of cortical neurons. We present a mathematical analysis for a class of these models. Contrasting with earlier theoretical work that has relied on methods from linear systems theory, we study the network's nonlinear dynamic properties that arise when the threshold nonlinearity of the neurons is taken into account. We show that such networks have stimulus-locked traveling pulse solutions that are appropriate for modeling the responses of direction selective cortical neurons. In addition, our analysis shows that outside a certain regime of stimulus speeds the stability of this solutions breaks down giving rise to another class of solutions that are characterized by specific spatio-temporal periodicity. This predicts that if direction selectivity in the cortex is mainly achieved by asymmetric lateral connections lurching activity waves might be observable in ensembles of direction selective cortical neurons within appropriate regimes of the stimulus speed.

## 1   Introduction

Classical models for the direction selectivity in the primary visual cortex have assumed feed-forward mechanisms, like multiplication or gating of afferent thalamo-cortical inputs (e.g. [1, 2, 3]), or linear spatio-temporal filtering followed by a nonlinear operation (e.g. [4, 5]). The existence of strong lateral connectivity has motivated modeling studies, which have shown that the properties of direction selective cortical neurons can also be accurately reproduced by recurrent neural network models with asymmetric lateral excitatory or inhibitory connections [6, 7]. Since these biophysically detailed models are not accessible for mathematical analysis, more simplified models appropriate for a mathematical analysis have been proposed. Such analysis was based on methods from linear systems theory by neglecting the nonlinear properties of the neurons [6, 8, 9]. The nonlinear dynamic phenomena resulting from the interplay between the recurrent connectivity and the nonlinear

threshold characteristics of the neurons have not been tractable in this theoretical framework.

In this paper we present a mathematical analysis that takes the nonlinear behavior of the individual neurons into account. We present the result of the analysis of such networks for two types of threshold nonlinearities, for which closed-form analytical solutions of the network dynamics can be derived. We show that such nonlinear networks have a class of form-stable solutions, in the following signified as *stimulus-locked traveling pulses*, which are suitable for modeling the activity of direction selective neurons. Contrary to networks with linear neurons, the stability of the traveling pulse solutions in the nonlinear network can break down giving raise to another class of solutions *(lurching activity waves)* that is characterized by spatio-temporal periodicity. Our mathematical analysis and simulations showed that recurrent models with biologically realistic degrees of direction selectivity typically also show transitions between traveling pulse and lurching solutions.

## 2   Basic model

Dynamic neural fields have been proposed to model the average behavior of a large ensembles of neurons [10, 11, 12]. The scalar neural activity distribution $u(x, t)$ characterizes the average activity at time $t$ of an ensemble of functionally similar neurons that code for the position $x$, where $x$ can be any abstract stimulus parameter. By the continuous approximation of biophysically discrete neuronal dynamics it is in some cases possible to treat the nonlinear neural dynamics analytically.

The field dynamics of neural activation variable $u(x, t)$ is described by:

$$\tau \frac{\partial u(x, t)}{\partial t} + u(x, t) = \int_\Omega w(x - x') f(u(x', t)) dx' + b(x, t). \tag{1}$$

This dynamics is essentially a leaky integrator with a total input on the right hand side, which includes a feedfoward input term $b(x, t)$ and a feedback term that integrates the recurrent contributions from other laterally connected neurons. The *interaction kernel* $w(x - x')$ characterizes the average synaptic connection strength between the neurons coding position $x'$ and the neurons coding position $x$. $f$ is the *activation function* of the neurons. This function is nonlinear and monotonically increasing, and introduces the nonlinearity that makes it difficult to analyze the network dynamics.

With a moving stimulus at constant velocity $v$, it is often convenient to transform the static coordinate to the moving frame by changing variable $\xi = x - vt$. Under the new frame, the stimulus is stationary. Let $U(\xi, t) = u(x - vt, t)$. The dynamics for $U$ reads

$$\tau \frac{\partial U(\xi, t)}{\partial t} - \tau v \frac{\partial U(\xi, t)}{\partial \xi} + U(\xi, t) = \int_\Omega w(\xi - \xi') f(U(\xi', t)) d\xi' + b(\xi). \tag{2}$$

A stationary solution in the moving frame has to satisfy the following equation:

$$-\tau v \frac{dU^*(\xi)}{d\xi} + U^*(\xi) = \int_\Omega w(\xi - \xi') f(U^*(\xi')) d\xi' + b(\xi). \tag{3}$$

$U^*(\xi)$ corresponds to a traveling pulse solution with velocity $v$ in the original static coordinate. Therefore the traveling pulse solution driven by the moving stimulus can be found by solving Eq. (3), and the stability of the traveling pulse can be studied by perturbing the stationary solution in Eq. (2).

The neural field dynamics Eq. (2) is a nonlinear integro-differential equation. In most cases an analytic treatment of such equations is impossible. In this paper, we consider two biologically inspired special cases, which can be analytically solved. For this purpose we consider only one-dimensional neural fields and assume that the nonlinear activation function $f$ is either a step function or a linear threshold function.

# 3 Step activation function

We first consider step activation function $f(z) = \Theta(z)$ where $\Theta(z) = 1$ when $z > 0$ and zero otherwise. This form of activation function approximates activities of neurons, which, by saturation, are either active or inactive. For the one-dimensional case, we assume that only a single stationary excited regime with $(U^*(\xi) > 0)$ exists that is located between the points $(\xi_1^*, \xi_2^*)$. Only neurons inside this regime contribute to the integral, and accordingly Eq. (3) can be simplified following [11]. The spatial shape $U^*(\xi)$ of the stationary solution obeys the ordinary differential equation

$$-\tau v \frac{dU^*(\xi)}{d\xi} + U^*(\xi) = W(\xi - \xi_1^*) - W(\xi - \xi_2^*) + b(\xi), \qquad (4)$$

where $W(z) \equiv \int_0^z w(x)dx$. The solution of the above equation can be found by treating the boundaries $\xi_1^*$ and $\xi_2^*$ as fixed parameters, and solving Eq. (4).

To facilitate notation we define an integral operator $O$ with parameter $\alpha \neq 0$ as

$$O[g(z); \alpha] = \int_{z_0}^z g(m)e^{(z-m)/\alpha}dm, \qquad (5)$$

where $z_0 = -\infty$ for $\alpha < 0$ and $z_0 = +\infty$ otherwise. Using this operator we define the two functions

$$F(z) = O[W(z); \tau v]/(-\tau v) \quad \text{and} \quad B(z) = O[b(z); \tau v]/(-\tau v).$$

The solution of Eq. (4) can be written with these functions in the form

$$U^*(\xi) = F(\xi - \xi_1^*) - F(\xi - \xi_2^*) + B(\xi). \qquad (6)$$

For the boundary points $U^*(\xi_1^*) = U^*(\xi_2^*) = 0$ must be satisfied, leading to the transcendent equation system

$$-F(0) + F(\xi_1^* - \xi_2^*) = B(\xi_1^*) \qquad (7)$$
$$F(0) - F(\xi_2^* - \xi_1^*) = B(\xi_2^*), \qquad (8)$$

from which $\xi_1^*$ and $\xi_2^*$ can be determined.

## 3.1 Stability of the traveling pulse solution

The stability of the traveling pulse solution can be analyzed by perturbing the stationary solution in the moving coordinate system. Let $\delta U(\xi, t)$ be a small perturbation of $U^*(\xi)$. The linearized perturbation dynamics reads

$$\tau \frac{\partial \delta U}{\partial t} - \tau v \frac{\partial \delta U}{\partial \xi} + \delta U(\xi, t) = -w(\xi - \xi_1^*)\,\delta\xi_1 + w(\xi - \xi_2^*)\,\delta\xi_2, \qquad (9)$$

where $\delta\xi_i$ ($i = 1, 2$) are the perturbations of the boundary points of the exited regime from the stationary values of $\xi_i^*$ with $U(\xi_i^* + \delta\xi_i, t) = 0$. However, $\delta\xi_i$ is not independent of $\delta U(\xi, t)$, and the dependence can be found by noting that

$$U(\xi_i, t) = U(\xi_i^* + \delta\xi_i, t) = U(\xi_i^*, t) + \frac{\partial U(\xi_i^*, t)}{\partial \xi}\delta\xi_i + O(\delta\xi_i^2) = 0.$$

Since $U(\xi_i^*, t) = \delta U(\xi_i^*, t)$ to the first order we have $\delta\xi_i = -\delta U(\xi_i^*, t)/c_i^*$, where $c_i^* \equiv dU^*(\xi_i)/d\xi$. Substituting this back into the perturbed dynamics, we have

$$\tau \frac{\partial \delta U}{\partial t} - \tau v \frac{\partial \delta U}{\partial \xi} + \delta U(\xi, t) = \frac{w(\xi - \xi_1^*)}{c_1^*}\delta U(\xi_1^*, t) - \frac{w(\xi - \xi_2^*)}{c_2^*}\delta U(\xi_2^*, t).$$

Substitute solution of the form $\delta U(\xi, t) = e^{\lambda t} Y(\xi)$ into the above dynamics. After some calculation, the eigenvalue equation for $\lambda$ reads

$$[G(0) - c_1^*(1 + \tau\lambda)][G(0) + c_2^*(1 + \tau\lambda)] = G(\xi_1^* - \xi_2^*, \lambda)G(\xi_2^* - \xi_1^*, \lambda), \qquad (10)$$

where function $G(\cdot)$ is defined as

$$G(z, \lambda) = O[w(z); \tau v/(1 + \tau\lambda)](1 + \tau\lambda)/(-\tau v).$$

From the transcendent Eq. (10), $\lambda$ can be found. The traveling pulse solution is asymptotically stable only if the real parts of all eigenvalues $\lambda$ are negative.

### 3.2 Simulation results of step activation function model

We use the following function

$$w(x) = a_e \exp(-k_e|x - x_0|) - a_i \exp(-k_i|x - x_0|)$$

as an example interaction kernel, numerically simulate the dynamics and compare the simulation results with the above mathematical analysis. The stimulus used is a moving bar with constant width and amplitude. The results are shown in the left (a-e) panels of Fig. (1). Panel (a) shows the speed tuning curve plotted as the dependence of the peak activity of the traveling pulse as function of the stimulus velocity $v$. The solid lines indicate the results from the numerical simulation and the dotted lines represent results from the analytical solution. Panel (b) shows the maximum real part of the eigenvalues obtained from Eq. (10). For small and large stimulus velocities maximum of the real parts of $\lambda$ becomes positive indicating a loss of stability of the form-stable solution. To verify this result we calculated the variability of the peak activity over time in simulation. Panel (c) shows the average variability as function of the stimulus velocity. At the velocities for which the eigenvalues indicate a loss of stability the variability of the amplitudes suddenly increases, consistent with our interpretation as a loss of the form stability of the solution.

An interesting observation is illustrated in panels (d) and (e) that show a color-coded plot of the space-time evolution of the activity. Panel (e) shows the propagation of the form-stable traveling pulse. Panel (d) shows the solution that arises when stability is lost. This solution is characterized by a spatio-temporal periodicity that is defined in the moving coordinate system by $U(y + mL_0, t + nT_0) = U(y, t)$, where $L_0$ and $T_0$ are constants that depend on the network dynamics. Solutions of similar type have been described before in spiking networks [13].

## 4 Linear threshold activation function

In this case, the activation function is taken to be $f(z) = [z]^+ = \max\{z, 0\}$. Cortical neurons typically operate far below the saturation level. The linear threshold activation function is thus more suitable to capture the properties of real neurons while still permitting a relatively simple theoretical analysis. We consider a ring network with periodic boundary conditions. The dynamics is given by

$$\tau\frac{\partial}{\partial t}m(\theta, t) + m(\theta, t) = \left[\int_{-\pi}^{\pi} \frac{d\theta'}{2\pi} w(\theta - \theta')m(\theta', t) + b(\theta, t)\right]^+. \qquad (11)$$

This network can be shown equivalent to the the standard one in Eq. (1) by changing variables and transforming stimulus. We chose this form because it simplifies the mathematical analysis of ring networks. Again, we consider a moving stimulus with velocity $v$ and analyze the network in the moving frame.

## 4.1 General solutions and stability analysis

Because the activation function has linear threshold characteristics, inside the excited regime for which the total input ($u(\theta, t) > 0$) is positive the system is linear. One approach to solve this dynamics is therefore to find the solutions to the differential equation assuming the boundaries of the excited regime are given. The conditions at the boundaries lead to a set of self-consistent equations for the solutions to satisfy, from which the boundaries can be determined.

By denoting activities in moving coordinates as $M(\theta - vt, t) = m(\theta, t)$, the dynamics can be written as:

$$\tau \frac{\partial}{\partial t} M(\theta, t) - \tau v \frac{\partial}{\partial \theta} M(\theta, t) + M(\theta, t) = \left[ \int_{-\pi}^{\pi} w(\theta - \theta') M(\theta', t) (2\pi)^{-1} d\theta' + B(\theta) \right]^{+}.$$

Supposing the excited regime is $\theta \in (\theta_1(t), \theta_2(t))$, we solve the dynamics by Fourier transforming the above equation in the spatial domain $[-\pi, \pi)$. Let

$$\hat{m}_n(t) = \int_{-\pi}^{\pi} M(\theta, t) e^{ik_n \theta} (2\pi)^{-1} d\theta \qquad \hat{w}_n = \int_{-\pi}^{\pi} w(\theta) e^{ik_n \theta} (2\pi)^{-1} d\theta$$

$$C_{nl} = \hat{w}_l \int_{\theta_1}^{\theta_2} e^{i(k_n - k_l)\theta} (2\pi)^{-1} d\theta \qquad \hat{b}_n = \int_{\theta_1}^{\theta_2} B(\theta) e^{ik_n \theta} (2\pi)^{-1} d\theta.$$

where $k_n = 0, \pm 1, ...$ is the frequency. The stationary solution in moving coordinates can then be written as

$$\hat{\mathbf{m}}^* = (I + i\tau v K - C)^{-1} \hat{\mathbf{b}}, \tag{12}$$

where matrix $K$ is defined as the diagonal matrix $K \equiv \text{diag}(k_n)$. The components of the vector $\hat{\mathbf{m}}$ are $\hat{m}_n$, and those of $\hat{\mathbf{b}}$ are $\hat{b}_n$. The above solution has to satisfy two boundary conditions, from which $\theta_1$ and $\theta_2$ can be determined.

Stability of this traveling pulse solution can be analyzed by linear perturbation. Note that perturbed boundaries points do not contribute to the linearized perturbed dynamics since $\delta \theta_i u^*(\theta_i) = 0 (i = 1, 2)$, where $u^*(\theta)$ is the total input at the stationary solution of the moving frame on right hand side of Eq. (11). Therefore, the linearized perturbation dynamics can be fully characterized by the perturbed Fourier modes with fixed boundaries. Hence, the stability of the traveling pulse solution is determined by the eigenvalues of matrix $A = -(I + i\tau v K - C)$. If the largest real part of eigenvalues of $A$ is negative, then the stimulus locking traveling pulse is stable.

## 4.2 Simplified linear threshold network

The general solution introduced above requires the solution of an equation system. In practice, the Fourier series have to be truncated in order to obtain a finite number of Fourier components at the expense of an approximation error. Next we consider a special simple model for which an exact solution can be found that contains only two Fourier components for the interaction kernel $w$ and the input $b$. For this model a closed form solution and stability analysis is presented, that at the same time provides insight in some rather general properties of linear threshold networks.

The interaction kernel and feedforward input are assumed to have the following form:

$$w(\theta) = J_0 + J_1 \cos(\theta + \beta) \qquad b(\theta) = C_0 - C_1 \cos(\theta) \tag{13}$$

This network was used by Hansel and Sompolinsky as model of cortical orientation selectivity [14]. However different from their network, we consider here an asymmetric interaction kernel $w(\theta)$ and a form-constant moving stimulus $b(\theta - vt)$.

Since the interaction kernel $w$ and input $b$ only involve first two Fourier components, the dynamics can be fully determined in terms of its order parameters defined by

$$r_0(t) = \int_{-\pi}^{\pi} \frac{d\theta'}{2\pi} m(\theta', t) \qquad r_1(t) = \int_{-\pi}^{\pi} \frac{d\theta'}{2\pi} m(\theta', t) e^{i(\theta - \Psi)} \qquad (14)$$

where phase variable $\Psi$ is to restrict $r_1(t)$ to being real. In terms of these two order parameters plus the phase variable, the stimulus-locked traveling pulse solution and its stability conditions can be expressed analytically. Due to space limitation, the detailed derivations are omitted here. We show the theoretical results in right five panels of Fig. (1) and compare them with numerical simulations.

Similar to the results of step function model, panel (A) shows the speed tuning curve plotted as values of order parameters $r_0$ and $r_1$ as function of different stimulus velocities $v$. Panel (B) shows the largest real part of the eigenvalues of a stability matrix that can be obtained by linearizing the order parameter dynamics around the stationary solution. Panel (C) shows the average variations as function of the stimulus velocity. The space-time evolution of the form-stable traveling pulse is shown in panel (E); the form-unstable lurching wave is shown in panel (D). Thus we found that lurching wave solution type arises very robustly for both types of threshold functions when the network achieved substantial direction selective behavior.

## 5    Conclusion

We have presented different methods for an analysis of the nonlinear dynamics of simple recurrent neural models for the direction selectivity of cortical neurons. Compared to earlier works, we have taken into account the essentially nonlinear effects that are introduced by the nonlinear threshold characteristics of the cortical neurons. The key result of our work is that such networks have a class of form-stable traveling pulse solutions that behave similar as the solutions of linear spatio-temporal filtering models within a certain regime of stimulus speeds. By the essential nonlinearity of the network, however, bifurcations can arise for which the traveling pulse solutions become unstable. We observed that in this case a new class of spatio-temporally periodic solutions ("lurching activity waves") arises. Since we found this solution type very frequently for networks with substantial direction selectivity our analysis predicts that such "lurching behavior" might be observable in visual cortex areas if, in fact, the direction selectivity is essentially based on asymmetric lateral connectivity.

**Acknowledgments**

We acknowledge helpful discussions with H.S. Seung and T. Poggio.

## References

[1] C Koch and T Poggio. The synaptic veto mechanism: does it underlie direction and orientation selectivity in the visual cortex. In D Rose and V G Dobson, editors, *Models of the Visual Cortex*, pages 15–34. John Wiley, 1989.

[2] J.P. van Santen and G. Sperling. Elaborated reichardt detectors. *J Opt Soc Am A*, 256:300–21, 1985.

[3] W. Reichardt. A principle for the evaluation of sensory information by the central nervous system, 1961.

[4] E. H. Adelson and J. R. Bergen. Spatiotemporal energy models for the perception of motion. *J Opt Soc Am A*, 256:284–99, 1985.

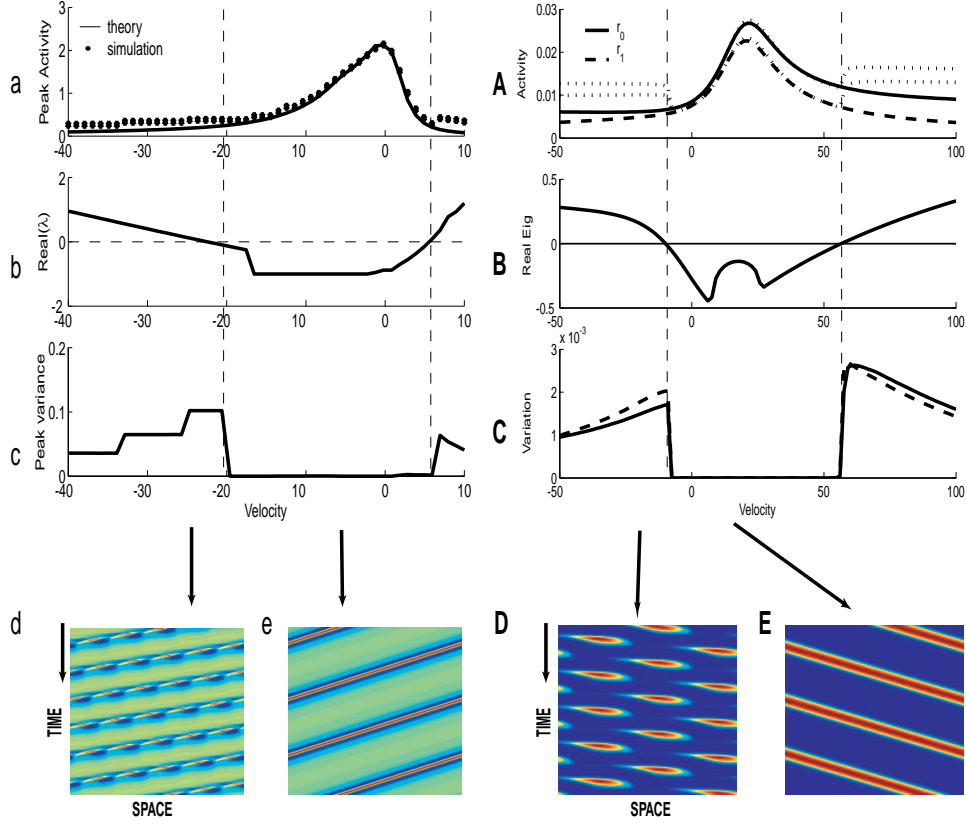

Figure 1: Traveling pulse solution and its stability in two classes of models. In the left side shown is the step activation function model, while the linear threshold model is drawn in the right. Panel (a) and (A) show the velocity tuning curves of the traveling pulse in terms of its peak activity in (a) or order parameters in (A). The solid lines indicate the results from calculation, and the dotted lines represents the results from simulaion. Panel (b) and (B) plot the largest real parts of eigenvalues of a stability matrix obtained from perturbed linear dynamics around the stationary solution. Outside certain range of stimulus velocities the largest real part of the eigenvalues become positive indicating a loss of stability of the form-stable solution. Panel (c) and (C) plots the average variations of peak activity, and order parameters $r_0$(blue curve) and $r_1$(green curve) respectively, over time during simulation. A nonzero variance signifies a loss of stability for traveling pulse solutions, which is consistent with eigenvalue analysis in Panel (b) and (B). A color coded plot of spatial-temporal evolution of the activity $u(x,t)$ is shown in panels (d) and (e), and $m(x,t)$ in (D) and (E). Panel (e) and (E) show the propagation of the form-stable peak over time; panel (d) and (D) show the lurching activity wave that arises when stability is lost. The interaction kernel used in step function model is $w(x) = a_e \exp(-k_e|x - x_0|) - a_i \exp(-k_i|x - x_0|)$ with $a_e = 1, a_i = 5, k_e = 0.42, k_i = 0.1$ and $x_0 = 3$. The stimulus is a moving bar with width $d = 10$ and amplitude $h = 2$. Parameters used in linear threshold model are $J_0 = -9.8, J_1 = 13.5, C_0 = C_1 = 0.05$ and $\tau = 0.01s$.

[5] A. B. Watson and A. J. Ahumada. Model of human visual-motion sensing. *J Opt Soc Am A*, 256:322–41, 1985.

[6] H. Suarez, C. Koch, and R. Douglas. Modeling direction selectivity of simple cells in striate visual cortex within the framework of the canonical microcircuit. *J Neurosci*, 15:6700–19, 1995.

[7] R. Maex and G. A. Orban. Model circuit of spiking neurons generating directional selectivity in simple cells. *J Neurophysiol*, 75:1515–45, 1996.

[8] P. Mineiro and D. Zipser. Analysis of direction selectivity arising from recurrent cortical interactions. *Neural Comput*, 10:353–71, 1998.

[9] S. P. Sabatini and F. Solari. An architectural hypothesis for direction selectivity in the visual cortex: the role of spatially asymmetric intracortical inhibition. *Biol Cybern*, 80:171–83, 1999.

[10] HR Wilson and JD Cowan. A mathematical theory of the functional dynamics of cortical and thalamic nervous tissue. *Kybernetik*, 13(2):55–80, 1973.

[11] S Amari. Dynamics of pattern formation in lateral-inhibition type neural fields. *Biol Cybern*, 27(2):77–87, 1977.

[12] E. Salinas and L.F. Abbott. A model of multiplicative neural responses in parietal cortex. *Proc. Natl. Acad. Sci. USA*, 93:11956–11961, 1996.

[13] D. Golomb and G. B. Ermentrout. Effects of delay on the type and velocity of travelling pulses in neuronal networks with spatially decaying connectivity. *Network*, 11:221–46, 2000.

[14] David Hansel and Haim Sompolinsky. Modeling feature selectivity in local cortical circuits. In C. Koch and I. Segev, editors, *Methods in Neuronal Modeling*, chapter 13, pages 499–567. MIT Press, Cambridge, Massachusetts, 1998.
